# Temporally Dependent Plasticity: An Information Theoretic Account

**Gal Chechik** and **Naftali Tishby**
School of Computer Science and Engineering
and the Interdisciplinary Center for Neural Computation
The Hebrew University, Jerusalem, Israel
{ggal,tishby}@cs.huji.ac.il

## Abstract

The paradigm of Hebbian learning has recently received a novel interpretation with the discovery of synaptic plasticity that depends on the relative timing of pre and post synaptic spikes. This paper derives a temporally dependent learning rule from the basic principle of mutual information maximization and studies its relation to the experimentally observed plasticity. We find that a supervised spike-dependent learning rule sharing similar structure with the experimentally observed plasticity increases mutual information to a stable near optimal level. Moreover, the analysis reveals how the temporal structure of time-dependent learning rules is determined by the temporal filter applied by neurons over their inputs. These results suggest experimental prediction as to the dependency of the learning rule on neuronal biophysical parameters

## 1 Introduction

Hebbian plasticity, the major paradigm for learning in computational neuroscience, was until a few years ago interpreted as learning by correlated neuronal activity. A series of studies have recently shown that changes in synaptic efficacies highly depend on the relative timing of the pre- and postsynaptic spikes, as the efficacy of a synapse between two excitatory neurons increases when the presynaptic spike precedes the postsynaptic one, but decreases otherwise [1-6]. The magnitude of these synaptic changes decays roughly exponentially as a function of the time difference between pre- and post synaptic spikes, with a time constant of few tens of milliseconds (results vary between studies, especially with regard to the synaptic depression component, compare e.g. [4] and [6]).

What could be the computational role of this delicate type of plasticity, sometimes termed *spike-timing dependent plasticity* (STDP) ? Several authors suggested answers for this question by modeling STDP and studying its effects on synaptic, neural and network dynamics. Importantly, STDP embodies an inherent competition between incoming inputs, and was shown to result in normalization of total incoming synaptic strength [7], maintain the irregularity of neuronal firing [8, 9],

and lead to the emergence of synchronous subpopulation firing in recurrent networks [10]. It may also play an important role in sequence learning [11, 12]. The dynamics of synaptic efficacies under the operation of STDP strongly depends on whether STDP is implemented additively (independent of the baseline synaptic value) or multiplicatively (where the change is proportional to the synaptic efficacy) [13].

This paper takes a different approach to the study of spike-dependent learning rules: while the above studies model STDP and study the model properties, we start by deriving a spike-dependent learning rule from first principles within a simple rate model and then compare it with the experimentally observed STDP. To derive our learning rule, we consider the principle of mutual information maximization. This idea, known as the *Infomax principle* [14], states that the goal of a neural network's learning procedure is to maximize the mutual information between its output and input. The current paper applies Infomax for a leaky integrator neuron with spiking inputs. The derivation suggests computational insights into the dependence of the temporal characteristics of STDP on biophysical parameters and shows that STDP may serve to maximize mutual information in a network of spiking neurons.

## 2   The Model

We study a network with $N$ input neurons $S_1..S_N$ firing spike trains, and a single output (target) neuron $Y$. At any point in time, the target neuron accumulates its inputs with some temporal filter $F$ due to voltage attenuation or synaptic transfer function

$$Y(t) = \sum_{i=1}^{N} W_i X_i(t) \quad ; \quad X_i(t) \equiv \int_{-\infty}^{t} F_\tau(t - t') S_i(t') dt' \tag{1}$$

where $W_i$ is the synaptic efficacy between the $i$th input neuron and the target neuron, $S_i(t) = \sum_{t_{spike}} \delta(t - t_{spike})$ is the $i$-th spike train and $\tau$ is the membrane time constant. The filter $F$ may be used to consider general synaptic transfer function and voltage decay effects, but is set here as an example to an exponential filter $F_\tau(x) \equiv exp(-x/\tau)$. The learning goal is to set the synaptic weights $W$ such that $M + 1$ uncorrelated patterns of input activity $\xi^\eta$ ($\eta = 0..M$) may be discriminated using the output. Each pattern determines the firing rates of the input neurons, thus $S$ is a noisy realization of $\xi$ due to the stochasticity of the point process. The input patterns are presented for periods of length $T$ (on the order of tens of milliseconds). At each period, a pattern $\xi^\eta$ is randomly chosen for presentation with probability $q_\eta$, where most of the patterns are rare ($\sum_{\eta=1}^{M} q_\eta \ll 1$) but $\xi^0$ is abundant and may be thought of as a background noisy pattern. It should be stressed that in our model information is coded in the non-stationary rates that underlie the input spike trains. As these rates are not observable, any learning must depends on the observable input spikes that realize those underlying rates.

## 3   Mutual Information Maximization

Let us focus on a single presentation period (omitting the notation of $t$), and look at the value of Y at the end of this period, $Y = \sum_{i=1}^{N} W_i X_i$, with $X_i \equiv \int_{-T}^{0} e^{t'/\tau} S_i(t') dt'$. Denoting by $f(Y)$ the p.d.f. of $Y$, the input-output mutual information [15] in this network is defined by

$$I(Y; \eta) = h(Y) - h(Y|\eta) \quad ; \quad h(Y) = -\int_{-\infty}^{\infty} f(y) log(f(y)) dx \tag{2}$$

where $h(Y)$ is the differential entropy of the $Y$ distribution, and $h(Y|\eta)$ is the differential entropy given that the network is presented with a known input pattern.

This mutual information measures how easy it is to decide which input pattern $\eta$ was presented to the network by observing the network's output $Y$.

To calculate the conditional entropy $h(Y|\eta)$ we use the assumption that input neurons fire independently and their number is large, thus the input of the target neuron when the network is presented with the pattern $\xi^\eta$ is normally distributed $f(Y|\eta) = N(\mu_\eta, \sigma_\eta^2)$ with mean $\mu_\eta = <WX^\eta>$ and variance $\sigma_\eta^2 = <(WX^\eta)(WX^\eta)^T> - <WX^\eta>^2$. The brackets denote averaging over the possible realizations of the inputs $X^\eta$ when the network is presented with the pattern $\xi^\eta$. To calculate the entropy of $Y$ we note that $f(Y)$ is a mixture of Gaussians, each resulting from the presentation of an input pattern and use the assumption $\sum_{\eta=1}^{M} q_\eta \ll 1$ to approximate the entropy. The details of this derivation are omitted due to space considerations and will be presented elsewhere. Differentiating the mutual information with regard to $W_i$ we obtain

$$\frac{\partial I(Y;\eta)}{\partial W_i} = +\sum_{\eta=1}^{M} q_\eta \left( Cov(Y, X_i^\eta) K_\eta^1 + E(X_i^\eta) K_\eta' \right) \tag{3}$$

$$- \sum_{\eta=1}^{M} q_\eta \left( Cov(Y, X_i^0) K_\eta^0 + E(X_i^0) K_\eta' \right)$$

with $\quad K_\eta^0 \equiv \dfrac{(\mu_\eta - \mu_0)^2}{\sigma_0^4} + \dfrac{\sigma_\eta^2 - \sigma_0^2}{\sigma_0^4}; \quad K_\eta^1 \equiv \dfrac{1}{\sigma_0} - \dfrac{1}{\sigma_\eta}; \quad K_\eta' \equiv \dfrac{\mu_\eta - \mu_0}{\sigma_0^2}.$

where $E(X_i^\eta)$ is the expected value of $X_i^\eta$ as averaged over presentation of the $\xi^\eta$ pattern . The general form of this complex gradient is simplified in the following sections together with a discussion of its use for biological learning.

The derived gradient may be used for a gradient ascent learning rule by repeatedly calculating the distribution moments $\mu_\eta, \sigma_\eta$ that depend on W, and updating the weights according to $\Delta W_i = \lambda \frac{\partial}{\partial W_i} I(Y;\eta)$. This learning rule climbs along the gradient and is bound to converge to a local maximum of the mutual information. Figure 1A plots the mutual information during the operation of the learning rule, showing that the network indeed reaches a (possibly local) mutual information maximum. Figure 1B depicts the changes in output distribution during learning, showing that it splits into two segregated bumps: one that corresponds to the $\xi^0$ pattern and another that corresponds to the rest of the patterns.

## 4 Learning In A Biological System

Aiming to obtain a spike-dependent biologically feasible learning rule that maximizes mutual information, we now turn to approximate the analytical rule derived above by a rule that can be implemented in biology. To this end, four steps are taken where each step corresponds to a biological constraint and its solution.

First, biological synapses are limited either to excitatory or inhibitory regimes. Since information is believed to be coded in the activity of excitatory neurons, we limit the weights $W$ to positive values.

Secondly, the $K$ terms are global functions of weights and input distributions since they depend on the distribution moments $\mu_\eta, \sigma_\eta$. To avoid this problem we approximate the learning rule by replacing $\{K_\eta^1, K_\eta^0, K_\eta'\}$ with constants $\{\lambda_\eta^1, \lambda_\eta^0, \lambda_\eta'\}$. These constants are set to optimal values, but remain fixed once they are set. We have found numerically that high performance (to be demonstrated in section 5) may be obtained for a wide regime of these constants.

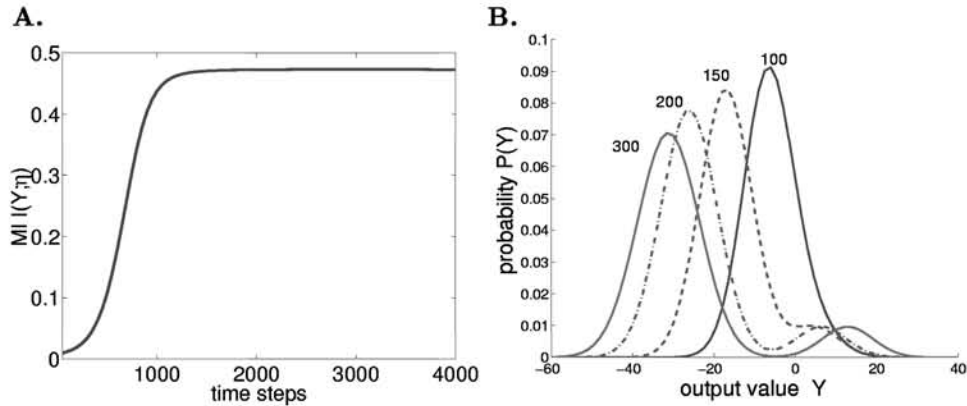

Figure 1: Mutual information and output distribution along learning with the gradient ascent learning rule (Eq. 3). All patterns were constructed by setting 10% of the input neurons to fire Poisson spike trains at $40Hz$, while the rest fire at $10Hz$. Poisson spike trains were simulated by discretizing time into 1 millisecond bins. Simulation parameters $\lambda = 1$ $M = 100$, $N = 1000$, $q_0 = 0.9$, $q_\eta = 0.001$, $T = 20msec$. **A.** Input-output mutual information **B.** Output distribution after 100,150,200 and 300 learning steps. Outputs segregate into two distinct bumps: one corresponds to the presentation of the $\xi^0$ pattern and the other corresponds to the rest of the patterns.

Thirdly, summation over patterns embodies a 'batch' mode of learning, requiring very large memory to average over multiple presentations. To implement an online learning rule, we replace summation over patterns by pattern-triggered learning. One should note that the analytical derivation yielded that summation in is performed over the rare patterns only (Eq. 3), thus pattern-triggered learning is naturally implemented by restricting learning to presentations of rare patterns[1].

Fourthly, the learning rule explicitly depends on $E(X)$ and $COV(Y, X)$ that are not observables of the model. We thus replace them by performing stochastic weighted averaging over spikes to yield a spike-dependent learning rule. In the case of inhomogeneous Poisson spike trains where input neurons fire independently, the covariance terms obeys $Cov(Y, X_i) = W_i E_{\tau/2}(X_i)$, where $E_\tau(X) = \int_{-\infty}^{t} e^{\frac{t'-t}{\tau}} E(S(t'))dt'$. The expectations $E(X_i^\eta)$ may be simply estimated by weighted averaging of the observed spikes $X_i^\eta$ that precede the learning moment. Estimating $E(X_i^0)$ is more difficult because, as stated above, learning should be triggered by the rare patterns only. Thus, $\xi^0$ spikes should have an effect only when a rare pattern $\xi^\eta$ is presented. A possible solution is to use the fact that $\xi^0$ is highly frequent, (and therefore spikes in the vicinity of a $\xi^\eta$ presentation are with high probability $\xi^0$ spikes), to average over spikes following a $\xi^\eta$ presentation for background activity estimation. These spikes can be temporally weighted in many ways: from the computational point of view it is beneficial to weigh spikes uniformly along time, but this may require long "memory" and is biologically improbable. We thus refrain from suggesting a specific weighting for background spikes, and obtain the following rule, that is

activated only when one of the rare patterns ($\xi^\eta, mem = 1..M$) is presented

$$\Delta W_i = \quad + \left( \lambda_\eta W_i \int_{-T}^{0} e^{\frac{t'-t}{\tau/2}} S_i(t')dt' + \lambda' \int_{-T}^{0} e^{\frac{t'-t}{\tau}} S_i(t')dt' \right) \qquad (4)$$
$$- \left( \lambda_0 W_i \int_{-\infty}^{\infty} f_1(t')S_i(t')dt' + \lambda' \int_{-\infty}^{\infty} f_2(t')S_i(t')dt' \right)$$

where $f_{1,2}(S(t'))$ denote the temporal weighting of $\xi^0$ spikes. It should be noted that this learning rule uses rare pattern presentations as an external ("supervised") learning signal. The general form of this learning rule and its performance are discussed in the next section.

# 5 Analyzing The Biologically Feasible Rule

## 5.1 Comparing performance

We have obtained a new spike-dependent learning rule that may be implemented in a biological system that approximates an information maximization learning rule. But how good are these approximations? Does learning with the biologically feasible learning rule increase mutual information? and to what level ? The curves in figure 2A compare the mutual information of the learning rule of Eq. 3 with that of Eq. 4, as traced in simulation of the learning process. Apparently, the approximated learning rule achieves fairly good performance compared to the optimal rule, and most of reduction in performance is due to limiting weights to positive values.

## 5.2 Interpreting the learning rule structure

The general form of the learning rule of Eq. 4 is pictorially presented in figure 2B, to allow us to inspect the main features of its structure. First, synaptic potentiation is temporally weighted in a manner that is determined by the same filter $F$ that the neuron applies over its inputs, but learning should apply an average of $F$ and $F^2$ ($\int^t F(t - t')S(t')dt'$ and $\int^t F^2(t - t')S(t')dt'$). The relative weighting of these two components was numerically estimated by simulating the optimal rule of Eq. 3 and was found to be on the same order of magnitude. Second, in our model synaptic depression is targeted at learning the underlying structure of background activity. Our analysis does not restrict the temporal weighting of the depression curve.

A major difference between the obtained rule and the experimentally observed learning rule is that in our rule learning is triggered by an external learning signal that corresponds to the presentation of rare patterns, while in the experimentally observed rule learning is triggered by the postsynaptic spike. The possible role of the postsynaptic spike is discussed in the following section.

# 6 Unsupervised Learning

By now we have considered a learning scenario that used external information telling whether the presented pattern is the background pattern or not, to decide whether learning should take place. When such learning signal is missing, it is tempting to use the postsynaptic spike (signaling the presence of an interesting input pattern) as a learning signal. This yields a learning procedure as in Eq. 4 except this time learning is triggered by postsynaptic spikes instead of an external signal. The resulting learning rule is similar to previous models of the experimentally observed STDP as [9, 13, 16]. However, this mechanism will effectively serve learning only

if the postsynaptic spikes co-occur with the presentation of a rare pattern. Such co-occurrence may be achieved by supplying short learning signals at the presence of the interesting patterns (e.g. by attentional mechanisms increasing neuronal excitability). This will induce learning such that later postsynaptic spikes will be triggered by the rare pattern presentation. These issues await further investigation.

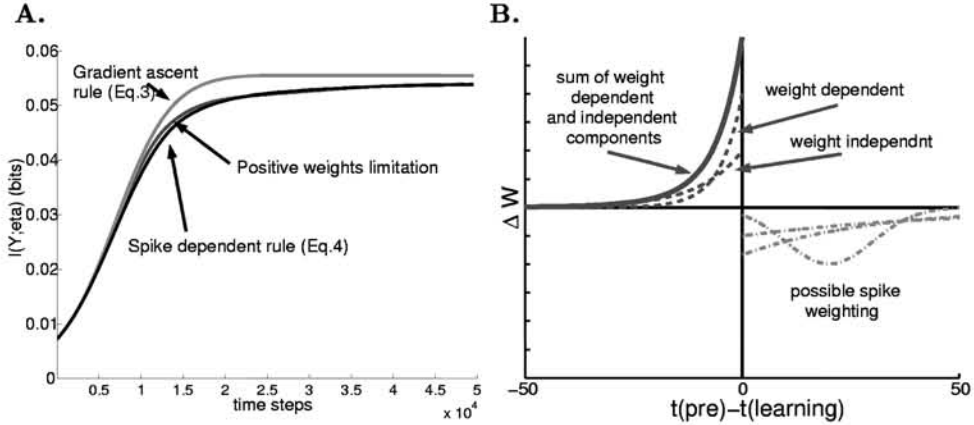

Figure 2: **A.** Comparing optimal (Eq. 3) and approximated (Eq. 4) learning rules. 10% of the input neurons of $\xi^\eta$ ($\eta > 0$) were set to fire at $40Hz$, while the rest fire at $5Hz$. $\xi^0$-neurons fire at $8Hz$ yielding similar average input as the $\xi^\eta$ patterns. Learning rates ratio for Eq. 4 were numerically searched for optimal value, yielding $\lambda_\eta = 0.15, \lambda_0 = 0.05$ for the arbitrary choice $\lambda' = 0.1$. Rest of parameters as in Fig 1 except $M = 20$, $N = 2000$. **B.** A pictorial representation of Eq. 4, plotting $\Delta W$ as a function of the time difference between the learning signal time $t$ and the input spike time $t_spike$. The potentiation curve (solid line) is the sum of two exponents with constants $\tau$ and $\frac{1}{2}\tau$ (dashed lines). The depression curve is not constrained by our derivation, thus several examples are brought (dot-dashed lines).

## 7   Discussion

In the framework of information maximization, we have derived a spike-dependent learning rule for a leaky integrator neuron. This learning rule achieves near optimal mutual information and can in principle be implemented in biological neurons. The analytical derivation of this rule allows to obtain insight into the learning rules observed experimentally in various preparations.

The most fundamental result is that time-dependent learning stems from the time-dependency of neuronal output on its inputs. In our model this is embodied in the filter $F$ which a neuron applies over its input spike trains. This filter is determined by the biophysical parameters of the neuron, namely its membrane leak, synaptic transfer functions and dendritic arbor structure. Our model thus yields direct experimental predictions for the way temporal characteristics of the potentiation learning curve are determined by the neuronal biophysical parameters. Namely, cells with larger membrane constants should exhibit longer synaptic potentiation time windows. Interestingly, the time window observed for STDP potentiation indeed fits the time windows of an AMPA channel and is also in agreement with cortical membrane time constants, as predicted by the current analysis [4, 6].

Several features of the theoretically derived rule may have similar functions in the experimentally observed rule: In our model synaptic weakening is targeted to learn the structure of the background activity. Both synaptic depression and potentiation in our model should be triggered by rare pattern presentation to allow near-optimal

mutual information. IN addition, synaptic changes should depend on the synaptic baseline value in a sub-linear manner. The experimental results in this regard are still unclear, but theoretical investigations show that this weight dependency has large effect on networks dynamics [13].

While the learning rule presented in Equation 4 assumes independent firing of input neurons, our derivation actually holds for a wider class of inputs. In the case of correlated inputs however, the learning rule involves cross-synaptic terms, which may be difficult to compute by biological neurons. As STDP is highly sensitive to synchronous inputs, it remains a most interesting question to investigate biologically-feasible approximations to an Infomax rule for time structured and synchronous inputs.

## Footnotes

*Work supported in part by a Human Frontier Science Project (HFSP) grant RG 0133/1998.

[1]In fact, learning rules where learning is also triggered by the presentation of the background pattern explicitly depend on the prior probabilities $q_\eta$, and thus are not robust to fluctuations in $q_\eta$. Since such fluctuations strongly reduce the mutual information obtained by these rules, we conclude that pattern-triggered learning should be triggered by the rare pattern only.

# References

[1] W.B. Levy and D. Steward. Temporal contiguity requirements for long-term associative potentiatio/depression in the hippocampus. *Neuroscience*, 8:791–797, 1983.

[2] D. Debanne, B.H. Gahwiler, and S.M. Thompson. Asynchronous pre- and post-synaptic activity induces associative long-term depression in area CA1 of the rat hippocampus in vitro. *Proc. Natl. Acad. Sci.*, 91:1148–1152, 1994.

[3] H. Markram, J. Lubke, M. Frotscher, and B. Sakmann. Regulation of synaptic efficacy by coincidence of postsynaptic aps and epsps. *Science*, 275(5297):213–215, 1997.

[4] L. Zhang, H.W.Tao, C.E. Holt, W.A. Harris, and M m. Poo. A critical window for cooperation and competition among developing retinotectal synapses. *Nature*, 395(3):37–44, 1998.

[5] Q. Bi and M m. Poo. Precise spike timing determines the direction and extent of synaptic modifications in cultured hippocampal neurons. *J. Neurosci.*, 18:10464–10472, 1999.

[6] D.E. Feldman. Timing based ltp and ltd at vertical inputs to layer II/III pryamidal cells in rat barrel cortex. *Neuron*, 27:45–56, 2000.

[7] R. Kempter, W. Gerstner, and J.L. van Hemmen. Hebbian learning and spiking neurons. *Phys. Rev. E.*, 59(4):4498–4514, 1999.

[8] L.F. Abbot and S. Song. Temporally asymmetric hebbian learning, spike timing and neural respons variability. In S.A. Solla and D.A. Cohen, editors, *Advances in Neural Information Processing Systems 11*, pages 69–75. MIT Press, 1999.

[9] S. Song, K.D. Miller, and L.F. Abbot. Competitive Hebbian learning through spike-timing dependent synaptic plasticity. *Nature Neuroscience*, pages 919–926, 2000.

[10] D. Horn, N. Levy, I. Meilijson, and E. Ruppin. Distributed synchrony of spiking neurons in a hebbian cell assembly. In S.A. Solla, T.K. Leen, and K.R. Muller, editors, *Advances in Neural Information Processing Systems 12*, pages 129–135, 2000.

[11] M.R. Mehta, M. Quirk, and M. Wilson. From hippocampus to v1: Effect of ltp on spatio-temporal dynamics of receptive fields. In J.M. Bower, editor, *Computational Neuroscience: Trends in Research 1999*. Elsevier, 1999.

[12] R. Rao and T. Sejnowski. Predictive sequence learning in recurrent neocortical circuits. In S.A. Solla, T.K. Leen, and K.R. Muller, editors, *Advances in Neural Information Processing Systems 12*, pages 164–170. MIT Press, 2000.

[13] J. Rubin, D. Lee, and H. Sompolinski. Equilibrium properties of temporally asymmetric hebbian plasticity. *Phys. Rev. D.*, In press, 2000.

[14] R. Linsker. Self-organization in a perceptual network. *Computer*, 21(3):105–117, 1988.

[15] C.E. Shannon. A mathematical theory of communication. *Bell Syst. Tech. J.*, 27:379–423, 1948.

[16] R. Kempter, W. Gerstner, and J.L. van Hemmen. Intrinsic stabilization of output rates by spike-time dependent hebbian learning. *Submitted*, 2000.
